# An Adaptive WTA using Floating Gate Technology

**W. Fritz Kruger, Paul Hasler, Bradley A. Minch, and Christof Koch**
California Institute of Technology
Pasadena, CA 91125
(818) 395 - 2812
stretch@klab.caltech.edu

## Abstract

We have designed, fabricated, and tested an adaptive Winner–Take–All (WTA) circuit based upon the classic WTA of Lazzaro, et al [1]. We have added a time dimension (adaptation) to this circuit to make the input derivative an important factor in winner selection. To accomplish this, we have modified the classic WTA circuit by adding floating gate transistors which slowly null their inputs over time. We present a simplified analysis and experimental data of this adaptive WTA fabricated in a standard CMOS $2\mu$m process.

## 1  Winner–Take–All Circuits

In a WTA network, each cell has one input and one output. For any set of inputs, the outputs will all be at zero except for the one which is from the cell with the maximum input. One way to accomplish this is by a global nonlinear inhibition coupled with a self-excitation term [2]. Each cell inhibits all others while exciting itself; thus a cell with even a slightly greater input than the others will excite itself up to its maximal state and inhibit the others down to their minimal states. The WTA function is important for many classical neural nets that involve competitive learning, vector quantization and feature mapping. The classic WTA network characterized by Lazzaro et. al. [1] is an elegant, simple circuit that shares just one common line among all cells of the network to propagate the inhibition.

Our motivation to add adaptation comes from the idea of saliency maps. Picture a saliency map as a large number of cells each of which encodes an analog value

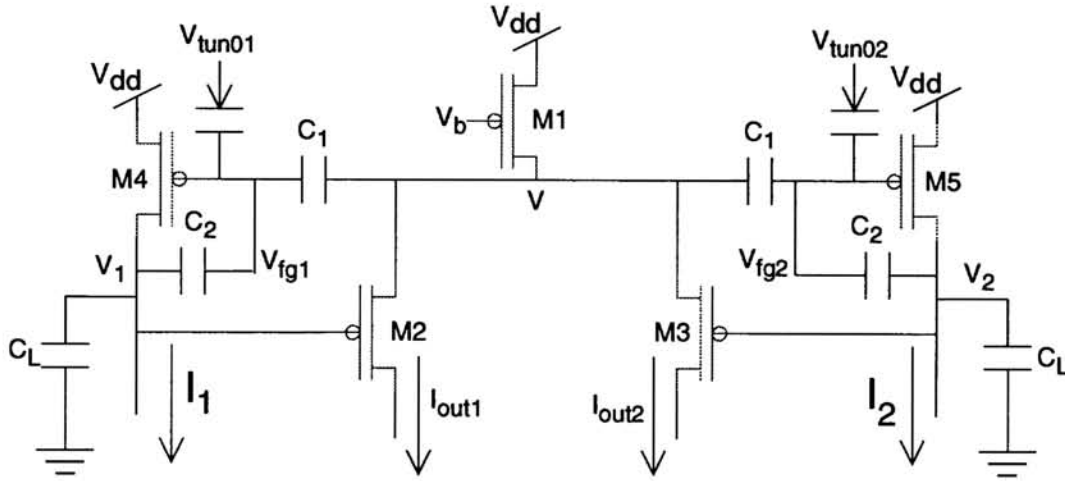

Figure 1: The circuit diagram of a two input winner-take-all circuit.

reflecting some measure of the importance (saliency) of its input. We would like to pay attention to the most salient cell, so we employ a WTA function to tell us where to look. But if the input doesn't change, we never look away from that one cell. We would like to introduce some concept of fatigue and refraction to each cell such that after winning for some time, it tires, allowing other cells to win, and then it must wait some time before it can win again. We call this circuit an adaptive WTA.

In this paper, we present an adaptive WTA based upon the classic WTA; Figure 1 shows a two-input, adaptive WTA circuit. The difference between the classic and adaptive WTA is that $M_4$ and $M_5$ are pFET single transistor synapses. A single transistor synapse [3] is either an nFET or pFET transistor with a floating gate and a tunneling junction. This enhancement results in the ability of each transistor to adapt to its input bias current. The adaptation is a result of the electron tunneling and hot-electron injection modifying the charge on the floating gate; equilibrium is established when the tunneling current equals the injection current. The circuit is devised in such a way that these are negative feedback mechanisms, consequently the output voltage will always return to the same steady state voltage determined by its bias current regardless of the DC input level. Like the autozeroing amplifier [4], the adaptive WTA is an example of a circuit where the adaptation occurs as a natural part of the circuit operation.

## 2  pFET hot-electron injection and electron tunneling

Before considering the behavior of the adaptive WTA, we will review the processes of electron tunneling and hot-electron injection in pFETs. In subthreshold operation, we can describe the channel current of a pFET ($I_p$) for a differential change in gate voltage, $\Delta V_g$, around a fixed bias current $I_{so}$, as $I_p = I_{so} \exp\left(-\frac{\kappa \Delta V_g}{U_T}\right)$ where $\kappa_p$ is the amount by which $\Delta V_g$ affects the surface potential of the pFET, and $U_T$ is $\frac{kT}{q}$. We will assume for this paper that all transistors are identical.

First, we consider electron tunneling. We start with the classic model of electron

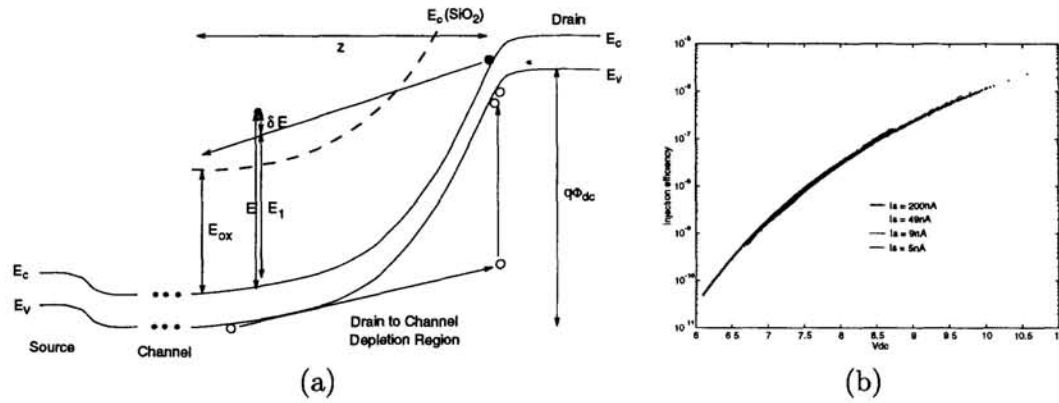

Figure 2: pFET Hot Electron Injection. (a) Band diagram of a subthreshold pFET transistor for favorable conditions for hot-electron injection. (b) Measured data of pFET injection efficiency versus the drain to channel voltage for four source currents. Injection efficiency is the ratio of injection current to source current. At $\Phi_{dc}$ equal to 8.2$V$, the injection efficiency increases a factor of $e$ for an increase $\Phi_{dc}$ of 250$mV$.

tunneling through a silicon - SiO$_2$ system [5]. As in the autozeroing amplifier [4], the tunneling current will be only a weak function for the voltage swing on the floating gate voltage through the region of subthreshold currents; therefore we will approximate the tunneling junction as a current source supplying $I_{tun0}$ current to the floating gate.

Second, we derive a simple model of pFET hot-electron injection. Figure 2a shows the band diagram of a pFET operating at bias conditions which are favorable for hot-electron injection. Hot-hole impact ionization creates electrons at the drain edge of the depletion region. These secondary electrons travel back into the channel region gaining energy as they go. When their energy exceeds that of the SiO$_2$ barrier, they can be injected through the oxide to the floating gate. The hole impact ionization current is proportional to the source current, and is an exponential function of the voltage drop from channel to drain ($\Phi_{dc}$). The injection current is proportional to the hole impact ionization current and is an exponential function of the voltage drop from channel to drain. We will neglect the dependence of the floating-gate voltage for a given source current and $\Phi_{dc}$ as we did in [4]. Figure 2b shows measured injection efficiency for several source currents, where injection efficiency is the ratio of the injection current to source current. The injection efficiency is independent of source current and is approximately linear over a 1 – 2$V$ swing in $\Phi_{dc}$; therefore we model the injection efficiency as proportional to $\exp\left(-\frac{\Delta\Phi_{dc}}{V_{inj}}\right)$ within that 1 to 2$V$ swing, where $V_{inj}$ is a measured device parameter which for our process is 250$mV$ at a bias $\Phi_{dc} = 8.2V$, and $\Delta\Phi_{dc}$ is the change in $\Phi_{dc}$ from the bias level. An increasing voltage input will increase the pFET surface potential by capacitive coupling to the floating gate. Increasing the pFET surface potential will increase the source current thereby decreasing $\Phi_{dc}$ for a fixed output voltage and lowering the injection efficiency.

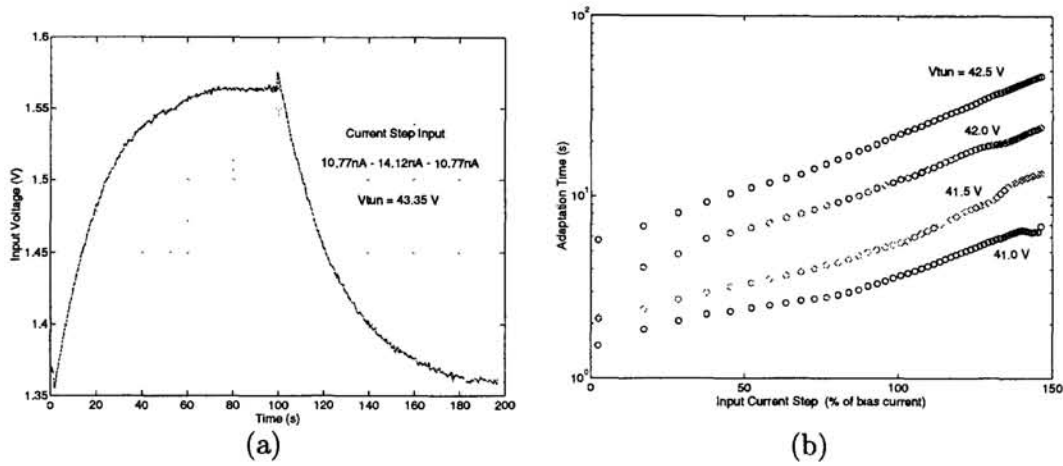

(a)          (b)

Figure 3: Illustration of the dynamics for the winning and losing input voltages. (a) Measured $V_1$ verses time due to an upgoing and a downgoing input current step. The initial input voltage change due to the input step is much smaller than the voltage change due to the adaptation. (b) Adaptation time of a losing input voltage for several tunneling voltages. The adaptation time is the time from the start of the input current step to the time the input voltage is within 10% of its steady state voltage. A larger tunneling current decreases the adaptation time by increasing the tunneling current supplied to the floating gate.

## 3   Two input Adaptive WTA

We will outline the general procedure to derive the general equations to describe the two input WTA shown in Fig. 1. We first observe that transistors $M_1$, $M_2$, and $M_3$ make up a differential pair. Regardless of any adaptation, the middle $V$ node and output currents are set by the input voltages ($V_1$ and $V_2$), which are set by the input currents, as in the classic WTA [1]. The dynamics for high frequency operation are also similar to the classic WTA circuit. Next, we can write the two Kirchhoff Current Law (KCL) equations at $V_1$ and $V_2$, which relate the change in $V_1$ and $V_2$ as a function of the two input currents and the floating gate voltages. Finally, we can write the two KCL equations at the two floating gates $V_{fg1}$ and $V_{fg2}$, which relates the changes in the floating gate voltages in terms of $V_1$ and $V_2$. This procedure is directly extendable to multiple inputs. A full analysis of these equations is very difficult and will be described in another paper.

For this discussion, we present a simplified analysis to develop the intuition of the circuit operation. At sufficiently high frequencies, the tunneling and injection currents do not adapt the floating gate voltages sufficiently fast to keep the input voltages at their steady state levels. At these frequencies, the adaptive WTA acts like the classic WTA circuit with one small difference. A change in the input voltages, $V_1$ or $V_2$ is linearly related to $V$ by the capacitive coupling ($\Delta V_1 = -\frac{C_1}{C_2}\Delta V$), where this relationship is exponential in the classic WTA. There is always some capacitance $C_2$, even if not explicitly drawn due to the overlap capacitance from the floating gate to drain. This property gives the designer the added freedom to modify the gain. We will assume the circuit operates in its intended operating regime where the floating gate transistors settle sufficiently fast such that their channel

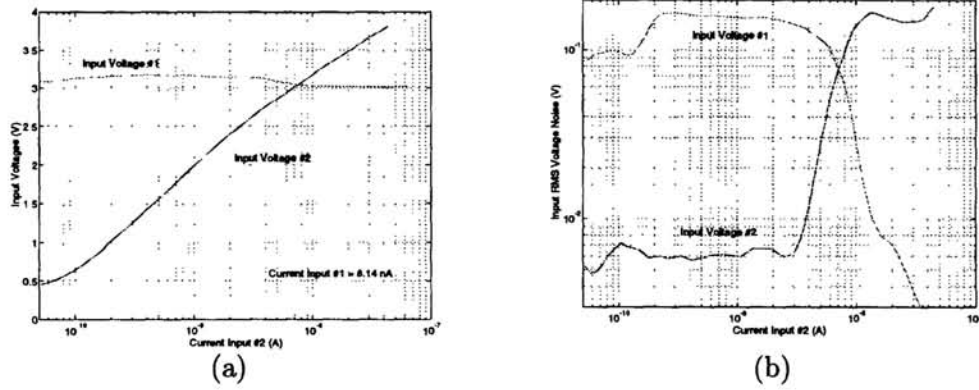

<center>(a)                                                              (b)</center>

Figure 4: Measured change in steady state input voltages as a function of bias current. (a) Change in the two steady state output voltages as a function of the bias current of the second input. The bias current of the first input was held fixed at $8.14nA$. (b) Change in the RMS noise of the two output voltages as a function of the bias current of the second input. The RMS noise is much higher for the losing input than for the winning input. Note that where the two bias currents cross roughly corresponds to the location where the RMS noise on the two input voltages is equal.

current equals the input currents

$$I_i = I_{so} \exp(-\frac{\kappa \Delta V_{fgi}}{U_T}) \rightarrow \frac{dI_i}{dt} = -I_i \frac{\kappa}{U_T} \frac{dV_{fgi}}{dt} \tag{1}$$

for all inputs indexed by $i$, but not necessarily fast enough for the floating gates to settle to their final steady state levels.

To develop some initial intuition, we shall begin by considering one half of the two input WTA: transistors $M_1$, $M_2$ and $M_4$ of Figure 1. First, we notice that $I_{out1}$ is equal to $I_b$ (the current through transistor $M_1$); note that this is not true for the multiple input case. By equating these two currents we get an equation for $V$ as $V = \kappa V_1 - \kappa V_b$, where we will assume that $V_b$ is a fixed bias voltage. Assuming the input current equals the current through $M_4$, $V_1$ obeys the equation

$$(\kappa C_1 + C_2)\frac{dV_1}{dt} = -\frac{C_T U_T}{\kappa I_1}\frac{dI_1}{dt} + I_{tun0}\left(\frac{I_1}{I_{so}}\exp(-\frac{\Delta V_1}{V_{inj}}) - 1\right) \tag{2}$$

where $C_T$ is the total capacitance connected to the floating gate. The steady state of (2) is

$$\Delta V_{in} = \frac{\kappa V_{inj}}{U_T} \ln\left(\frac{I_1}{I_{so}}\right) \tag{3}$$

which is exactly the same expression for each input in a multiple input WTA. We get a linear differential equation by making the substitution $X = \exp(\frac{\Delta V_1}{V_{inj}})$ [4], and we get similar solutions to the behavior of the autozeroing amplifier. Figure 3a shows measured data for an upgoing and a downgoing current step. The input current change results in an initial fast change in the input voltage, and the input voltage then adapts to its steady state voltage which is a much greater voltage change. From the voltage difference between the steady states, we get that $V_{inj}$ is roughly $500mV$.

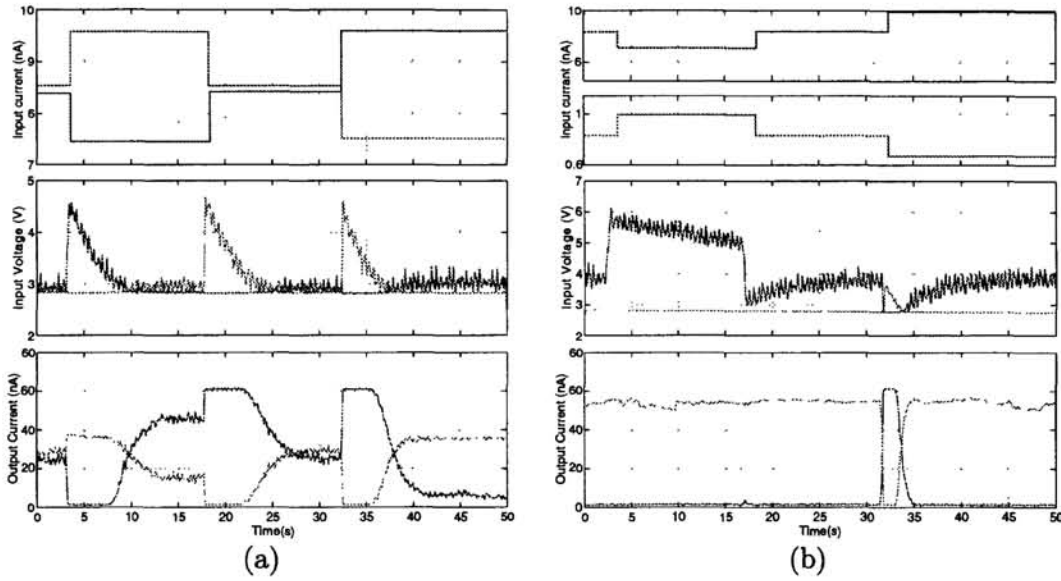

Figure 5: Experimental time traces measurements of the output current and voltage for small differential input current steps. (a) Time traces for small differential current steps around nearly identical bias currents of $8.6nA$. (b) Time traces for small differential current steps around two different bias currents of $8.7nA$ and $0.88nA$. In the classic WTA, the output currents would show no response to the input current steps.

Returning to the two input case, we get two floating gate equations by assuming that the currents through $M_4$ and $M_5$ are equal to their respective input currents and writing the KCL equations at each floating gate. If $V_1$ and $V_2$ do not cross each other in the circuit operation, then one can easily solve these KCL equations. Assume without loss of generality that $V_1$ is the winning voltage; which implies that $\Delta V = \kappa \Delta V_1$. The initial input voltage change before the floating gate adaptation due to a step in the two input currents of $I_1^- \to I_1^+$ and $I_2^- \to I_2^+$ is

$$\Delta V_1 = \frac{C_T}{\kappa C_1} \ln \left( \frac{I_1^+}{I_1^-} \right), \Delta V_2 \approx \frac{C_T}{C_2} \ln \left( \frac{I_1^-}{I_1^+} \frac{I_2^+}{I_2^-} \right) \qquad (4)$$

for $C_2$ much less than $\kappa C_1$. In this case, $V_1$ moves on the order of the floating gate voltage change, but $V_2$ moves on the order of the floating gate change amplified up by $\frac{C_T}{C_2}$. The response of $\Delta V_1$ is governed by an identical equation to (2) of the earlier half-analysis, and therefore results in a small change in $V_1$. Also, any perturbation of $V$ is only slightly amplified at $V_1$ due to the feedback; therefore any noise at $V$ will only be slightly amplified into $V_1$. The restoration of $V_2$ is much quicker than the $V_1$ node if $C_2$ is much less than $\kappa C_1$; therefore after the initial input step, one can safely assume that $V$ is nearly constant. The voltage at $V$ is amplified by $-\frac{C_1}{C_2}$ at $V_2$; therefore any noise at $V$ is amplified at the losing voltage, but not at the winning voltage as the data in Fig. 4b shows. The losing dynamics are identical to the step response of an autozeroing amplifier [4]. Figure 3b shows the variation of the adaptation time verses the percent input current change for several values of tunneling voltages.

The main difficulty in exactly solving these KCL equations is the point in the dynamics where $V_1$ crosses $V_2$, since the behavior changes when the signals move

through the crossover point. If we get more than a sufficient $V_1$ decrease to reach the starting $V_2$ equilibrium, then the rest of the input change is manifested by an increase in $V_2$. If the voltage $V_2$ crosses the voltage $V_1$, then $V$ will be set by the new steady state, and $V_1$ is governed by losing dynamics until $V_1 \approx V_2$. At this point $V_1$ is nearly constant and $V_2$ is governed by losing dynamics. This analysis is directly extendible to arbitrary number of inputs.

Figure 5 shows some characteristic traces from the two-input circuit. Recall that the winning node is that with the lowest voltage, which is reflected in its corresponding high output current. In Fig. 5a, we see that as an input step is applied, the output current jumps and then begins to adapt to a steady state value. When the inputs are nearly equal, the steady state outputs are nearly equal; but when the inputs are different, the steady state output is greater for the cell with the lesser input. In general, the input current change that is the largest after reaching the previous equilibrium becomes the new equilibrium. This additional decrease in $V_1$ would lead to an amplified increase in the other voltage since the losing stage roughly looks like an autozeroing amplifier with the common node as the input terminal. The extent to which the inputs do not equal this largest input is manifested as a proportionally larger input voltage. The other voltage would return to equilibrium by slowly, linearly decreasing in voltage due to the tunneling current. This process will continue until $V_1$ equals $V_2$. Note in general that the inputs with lower bias currents have a slight starting advantage over the inputs with higher bias currents.

Figure 5b illustrates the advantage of the adaptive WTA over the classic WTA. In the classic WTA, the output voltage and current would not change throughout the experiment, but the adaptive WTA responds to changes in the input. The second input step does not evoke a response because there was not enough time to adapt to steady state after the previous step; but the next step immediately causes it to win. Also note in both of these traces that the noise is very large in the loosing node and small in the winner because of the gain differences (see Figure 4b).

# References

[1] J. Lazzaro, S. Ryckebusch, M.A. Mahowald, and C.A. Mead "Winner–Take–All Networks of *O(N)* Complexity", *NIPS 1* Morgan Kaufmann Publishers, San Mateo, CA, 1989, pp 703 - 711.

[2] Grossberg S. "Adaptive Pattern Classification and Universal Recoding: I. Parallel Development and Coding of Neural Feature Detectors." *Biological Cybernetics* vol. 23, 121-134, 1988.

[3] P. Hasler, C. Diorio, B. A. Minch, and C. Mead, "Single Transistor Learning Synapses", *NIPS 7*, MIT Press, 1995, 817-824. *Also at http://www.pcmp.caltech.edu/anaprose/paul.*

[4] P. Hasler, B. A. Minch, C. Diorio, and C. Mead, "An autozeroing amplifier using pFET Hot-Electron Injection", *ISCAS*, Atlanta, 1996, III-325 - III-328. *Also at http://www.pcmp.caltech.edu/anaprose/paul.*

[5] M. Lenzlinger and E. H. Snow (1969), "Fowler-Nordheim tunneling into thermally grown $SiO_2$," *J. Appl. Phys.*, vol. 40, pp. 278-283, 1969.